# Continuous-Time Regression Models for Longitudinal Networks

**Duy Q. Vu**
Department of Statistics
Pennsylvania State University
University Park, PA 16802
dqv100@stat.psu.edu

**Arthur U. Asuncion**[*]
Department of Computer Science
University of California, Irvine
Irvine, CA 92697
asuncion@ics.uci.edu

**David R. Hunter**
Department of Statistics
Pennsylvania State University
University Park, PA 16802
dhunter@stat.psu.edu

**Padhraic Smyth**
Department of Computer Science
University of California, Irvine
Irvine, CA 92697
smyth@ics.uci.edu

## Abstract

The development of statistical models for continuous-time longitudinal network data is of increasing interest in machine learning and social science. Leveraging ideas from survival and event history analysis, we introduce a continuous-time regression modeling framework for network event data that can incorporate both time-dependent network statistics and time-varying regression coefficients. We also develop an efficient inference scheme that allows our approach to scale to large networks. On synthetic and real-world data, empirical results demonstrate that the proposed inference approach can accurately estimate the coefficients of the regression model, which is useful for interpreting the evolution of the network; furthermore, the learned model has systematically better predictive performance compared to standard baseline methods.

## 1  Introduction

The analysis of the structure and evolution of network data is an increasingly important task in a variety of disciplines, including biology and engineering. The emergence and growth of large-scale online social networks also provides motivation for the development of longitudinal models for networks over time. While in many cases the data for an evolving network are recorded on a continuous time scale, a common approach is to analyze "snapshot" data (also known as collapsed panel data), where multiple cross-sectional snapshots of the network are recorded at discrete time points. Various statistical frameworks have been previously proposed for discrete snapshot data, including dynamic versions of exponential random graph models [1, 2, 3] as well as dynamic block models and matrix factorization methods [4, 5]. In contrast, there is relatively little work to date on continuous-time models for large-scale longitudinal networks.

In this paper, we propose a general regression-based modeling framework for continuous-time network event data. Our methods are inspired by survival and event history analysis [6, 7]; specifically, we employ multivariate counting processes to model the edge dynamics of the network. Building on recent work in this context [8, 9], we use both multiplicative and additive intensity functions that allow for the incorporation of arbitrary time-dependent network statistics; furthermore, we consider

---

[*]current affiliation: Google Inc.

time-varying regression coefficients for the additive approach. The additive form in particular enables us to develop an efficient online inference scheme for estimating the time-varying coefficients of the model, allowing the approach to scale to large networks. On synthetic and real-world data, we show that the proposed scheme accurately estimates these coefficients and that the learned model is useful for both interpreting the evolution of the network and predicting future network events.

The specific contributions of this paper are: (1) We formulate a continuous-time regression model for longitudinal network data with time-dependent statistics (and time-varying coefficients for the additive form); (2) we develop an accurate and efficient inference scheme for estimating the regression coefficients; and (3) we perform an experimental analysis on real-world longitudinal networks and demonstrate that the proposed framework is useful in terms of prediction and interpretability.

The next section introduces the general regression framework and the associated inference scheme is described in detail in Section 3. Section 4 describes the experimental results on synthetic and real-world networks. Finally, we discuss related work and conclude with future research directions.

## 2 Regression models for continuous-time network data

Below we introduce multiplicative and additive regression models for the edge formation process in a longitudinal network. We also describe non-recurrent event models and give examples of time-dependent statistics in this context.

### 2.1 General framework

Assume in our network that nodes arrive according to some stochastic process and directed edges among these nodes are created over time. Given the ordered pair $(i, j)$ of nodes in the network at time $t$, let $N_{ij}(t)$ be a counting process denoting the number of edges from $i$ to $j$ up to time $t$. In this paper, each $N_{ij}(t)$ will equal zero or one, though this can be generalized. Combining the individual counting processes of all potential edges gives a multivariate counting process $\mathbf{N}(t) = (N_{ij}(t) : i, j \in \{1, \ldots n\}, i \neq j)$; we make no assumption about the independence of individual edge counting processes. (See [7] for an overview of counting processes.) We do not consider an edge dissolution process in this paper, although in theory it is possible to do so by placing a second counting process on each edge for dissolution events. (See [10, 3] for different examples of formation–dissolution process models.) As proposed in [9], we model the multivariate counting process via the Doob-Meyer decomposition [7],

$$\mathbf{N}(t) = \int_0^t \boldsymbol{\lambda}(s) \, ds + \mathbf{M}(t), \tag{1}$$

where essentially $\boldsymbol{\lambda}(t)$ and $\mathbf{M}(t)$ may be viewed as the (deterministic) signal and (martingale) noise, respectively. To model the so-called intensity process $\boldsymbol{\lambda}(t)$, we denote the entire past of the network, up to but not including time $t$, by $\mathbf{H}_{t-}$ and consider for each potential directed edge $(i, j)$ two possible intensity forms, the multiplicative Cox and the additive Aalen functions [7], respectively:

$$\lambda_{ij}(t|\mathbf{H}_{t-}) = Y_{ij}(t)\alpha_0(t)\exp\left[\boldsymbol{\beta}^\top \mathbf{s}(i, j, t)\right]; \tag{2}$$

$$\lambda_{ij}(t|\mathbf{H}_{t-}) = Y_{ij}(t)\left[\beta_0(t) + \boldsymbol{\beta}(t)^\top \mathbf{s}(i, j, t)\right], \tag{3}$$

where the "at risk" indicator function $Y_{ij}(t)$ equals one if and only if $(i, j)$ could form an edge at time $t$, a concept whose interpretation is determined by the context (e.g., see Section 2.2). In equations (2) and (3), $\mathbf{s}(i, j, t)$ is a vector of $p$ statistics for directed edge $(i, j)$ constructed based on $\mathbf{H}_{t-}$; examples of these statistics are given in Section 2.2. In each of the two models, the intensity process depends on a linear combination of the coefficients $\boldsymbol{\beta}$, which can be time-varying in the additive Aalen formulation. When all elements of $s_k(i, j, t)$ equal zero, we obtain the baseline hazards $\alpha_0(t)$ and $\beta_0(t)$.

The two intensity forms above, the Cox and Aalen, each have their respective strengths (e.g., see [7, chapter 4]). In particular, the coefficients of the Aalen model are quite easy to estimate via linear regression, unlike the Cox model. We leverage this computational advantage to develop an efficient inference algorithm for the Aalen model later in this paper. On the other hand, the Cox model forces the hazard function to be non-negative, while the Aalen model does not—however, in our experiments on both simulated and real-world data we did not encounter any issues with negative hazard functions when using the Aalen model.

## 2.2 Non-recurrent event models for network formation processes

If $t_i^{arr}$ and $t_j^{arr}$ are the arrival times of nodes $i$ and $j$, then the risk indicator of equations (2) and (3) is $Y_{ij}(t) = I\big(\max(t_i^{arr}, t_j^{arr}) < t \le t_{e_{ij}}\big)$. The time $t_{e_{ij}}$ of directed edge $(i, j)$ is taken to be $+\infty$ if the edge is never formed during the observation time. The reason for the upper bound $t_{e_{ij}}$ is that the counting process is non-recurrent; i.e., formation of an edge means that it can never occur again.

The network statistics $\mathbf{s}(i, j, t)$ of equations (2) and (3), corresponding to the ordered pair $(i, j)$, can be time-invariant (such as gender match) or time-dependent (such as the number of two-paths from $i$ to $j$ just before time $t$). Since it has been found empirically that most new edges in social networks are created between nodes separated by two hops [11], we limit our statistics to the following:

1. Out-degree of sender $i$: $s_1(i, j, t) = \sum_{h \in V, h \ne i} N_{ih}(t^-)$
2. In-degree of sender $i$: $s_2(i, j, t) = \sum_{h \in V, h \ne i} N_{hi}(t^-)$
3. Out-degree of receiver $j$: $s_3(i, j, t) = \sum_{h \in V, h \ne j} N_{jh}(t^-)$
4. In-degree of receiver $j$: $s_4(i, j, t) = \sum_{h \in V, h \ne j} N_{hj}(t^-)$
5. Reciprocity: $s_5(i, j, t) = N_{ji}(t^-)$
6. Transitivity: $s_6(i, j, t) = \sum_{h \in V, h \ne i, j} N_{ih}(t^-) N_{hj}(t^-)$
7. Shared contactees: $s_7(i, j, t) = \sum_{h \in V, h \ne i, j} N_{ih}(t^-) N_{jh}(t^-)$
8. Triangle closure: $s_8(i, j, t) = \sum_{h \in V, h \ne i, j} N_{hi}(t^-) N_{jh}(t^-)$
9. Shared contacters: $s_9(i, j, t) = \sum_{h \in V, h \ne i, j} N_{hi}(t^-) N_{hj}(t^-)$

Here $N_{ji}(t^-)$ denotes the value of the counting process $(i, j)$ right before time $t$. While this paper focuses on the non-recurrent setting for simplicity, one can also develop recurrent models using this framework, by capturing an alternative set of statistics specialized for the recurrent case [8, 12, 9]. Such models are useful for data where interaction edges occur multiple times (e.g., email data).

# 3 Inference techniques

In this section, we describe algorithms for estimating the coefficients of the multiplicative Cox and additive Aalen models. We also discuss an efficient online inference technique for the Aalen model.

## 3.1 Estimation for the Cox model

Recent work has posited Cox models similar to (2) with the goal of estimating general network effects [8, 12] or citation network effects [9]. Typically, $\alpha_0(t)$ is considered a nuisance parameter, and estimation for $\boldsymbol{\beta}$ proceeds by maximization of the so-called partial likelihood of Cox [13]:

$$L(\boldsymbol{\beta}) = \prod_{e=1}^{m} \frac{\exp\big(\boldsymbol{\beta}^\top \mathbf{s}(i_e, j_e, t_e)\big)}{\sum_{i=1}^{n} \sum_{j \ne i} Y_{ij}(t_e) \exp\big(\boldsymbol{\beta}^\top \mathbf{s}(i, j, t_e)\big)}, \tag{4}$$

where $m$ is the number of edge formation events, and $t_e$, $i_e$, and $j_e$ are the time, sender, and receiver of the $e$th event. In this paper, maximization is performed via the Newton-Raphson algorithm. The covariance matrix of $\hat{\boldsymbol{\beta}}$ is estimated as the inverse of the negative Hessian matrix of the last iteration.

We use the caching method of [9] to compute the likelihood, the score vector, and the Hessian matrix more efficiently. We will illustrate this method through the computation of the likelihood, where the most expensive computation is for the denominator

$$\kappa(t_e) = \sum_{i=1}^{n} \sum_{j \ne i} Y_{ij}(t_e) \exp\big(\boldsymbol{\beta}^\top \mathbf{s}(i, j, t_e)\big). \tag{5}$$

For models such as the one in Section 2.2, a naïve update for $\kappa(t_e)$ needs $O(pn^2)$ operations, where $n$ is the the current number of nodes. A naïve calculation of $\log L(\boldsymbol{\beta})$ needs $O(mpn^2)$ operations (where $m$ is the number of edge events), which is costly since $m$ and $n$ may be large. Calculations of the score vector and Hessian matrix are similar, though they involve higher exponents of $p$.

Alternatively, as in [9], we may simply write $\kappa(t_e) = \kappa(t_{e-1}) + \Delta\kappa(t_e)$, where $\Delta\kappa(t_e)$ entails all of the possible changes that occur during the time interval $[t_{e-1}, t_e)$. Since we assume in this paper that edges do not dissolve, it is necessary to keep track only of the group of edges whose covariates change during this interval, which we call $U_{e-1}$, and those that first become at risk during this interval, which we call $C_{e-1}$. These groups of edges may be cached in memory during an initialization step; then, subsequent calculations of $\Delta\kappa(t_e)$ are simple functions of the values of $\mathbf{s}(i, j, t_{e-1})$ and $\mathbf{s}(i, j, t_e)$ for $(i, j)$ in these two groups (for $C_{e-1}$, only the time $t_e$ statistic is relevant).

The number of edges cached at each time step tends to be small, generally $O(n)$ because our network statistics $\mathbf{s}$ are limited to those based on node degrees and two-paths. This leads to substantial computational savings; since we must still initialize $\kappa(t_1)$, the total computational complexity of each Newton-Raphson iteration is $O(p^2 n^2 + m(p^2 n + p^3))$.

## 3.2 Estimation for the Aalen model

Inference in model (3) proceeds not for the $\beta_k$ parameters directly but rather for their time-integrals

$$B_k(t) = \int_0^t \beta_k(s)ds. \tag{6}$$

The reason for this is that $\mathbf{B}(t) = [B_1(t), \dots, B_p(t)]$ may be estimated straightforwardly using a procedure akin to simple least squares [7]: First, let us impose some ordering on the $n(n-1)$ possible ordered pairs $(i, j)$ of nodes. Take $\mathbf{W}(t)$ to be the $n(n-1) \times p$ matrix whose $(i,j)$th row equals $Y_{ij}(t)\mathbf{s}(i, j, t)^\top$. Then

$$\hat{\mathbf{B}}(t) = \int_0^t J(s)\mathbf{W}^-(s)d\mathbf{N}(s) = \sum_{t_e \leq t} J(t_e)\mathbf{W}^-(t_e)\Delta\mathbf{N}(t_e) \tag{7}$$

is the estimator of $\mathbf{B}(t)$, where the multivariate counting process $\mathbf{N}(t_e)$ uses the same ordering of its $n(n-1)$ entries as the $\mathbf{W}(t)$ matrix,

$$\mathbf{W}^-(t) = \left[\mathbf{W}(t)^\top \mathbf{W}(t)\right]^{-1}\mathbf{W}(t)^\top,$$

and $J(t)$ is the indicator that $\mathbf{W}(t)$ has full column rank, where we take $J(t)\mathbf{W}^-(t) = \mathbf{0}$ whenever $\mathbf{W}(t)$ does not have full column rank. As with typical least squares, a covariance matrix for these $\hat{\mathbf{B}}(t)$ may also be estimated [7]; we give a formula for this matrix in equation (11). If estimates of $\beta_k(t)$ are desired for the sake of interpretability, a kernel smoothing method may be used:

$$\hat{\beta}_k(t) = \frac{1}{b}\sum_{t_e} K\left(\frac{t - t_e}{b}\right)\Delta\hat{B}_k(t_e), \tag{8}$$

where $b$ is the bandwidth parameter, $\Delta\hat{B}_k(t_e) = \hat{B}_k(t_e) - \hat{B}_k(t_{e-1})$, and $K$ is a bounded kernel function with compact support $[-1, 1]$ such as the Epanechnikov kernel.

## 3.3 Online inference for the Aalen model

Similar to the caching method for the Cox model in Section 3.1, it is possible to streamline the computations for estimating the integrated Aalen model coefficients $\mathbf{B}(t)$. First, we rewrite (7) as

$$\hat{\mathbf{B}}(t) = \sum_{t_e \leq t} J(t_e)\left[\mathbf{W}(t_e)^\top \mathbf{W}(t_e)\right]^{-1}\mathbf{W}(t_e)^\top \Delta\mathbf{N}(t_e) = \sum_{t_e \leq t} \mathbf{A}^{-1}(t_e)\mathbf{W}(t_e)^\top \Delta\mathbf{N}(t_e), \tag{9}$$

where $\mathbf{A}(t_e) = \mathbf{W}(t_e)^\top \mathbf{W}(t_e)$ and $J(t_e)$ is omitted because for large network data sets and for reasonable choices of starting observation times, the covariate matrix is always of full rank. The computation of $\mathbf{W}(t_e)^\top \Delta\mathbf{N}(t_e)$ is simple because $\Delta\mathbf{N}(t_e)$ consists of all zeros except for a single entry equal to one. The most expensive computation is to update the $(p+1) \times (p+1)$ matrix $\mathbf{A}(t_e)$ at every event time $t_e$; inverting $\mathbf{A}(t_e)$ is not expensive since $p$ is relatively small.

Using $U_{e-1}$ and $C_{e-1}$ as in Section 3.1, the component $(k, l)$ of the matrix $\mathbf{A}(t_e)$ corresponding to covariates $k$ and $l$ can be written as $A_{kl}(t_e) = A_{kl}(t_{e-1}) + \Delta A_{kl}(t_{e-1})$, where

$$\Delta A_{kl}(t_{e-1}) = -\sum_{(i,j)\in U_{e-1}} W_{ijk}(t_{e-1})W_{ijl}(t_{e-1}) + \sum_{(i,j)\in U_{e-1}\cup C_{e-1}} W_{ijk}(t_e)W_{ijl}(t_e). \tag{10}$$

For models such as the one presented in Section 2.2, if $n$ is the current number of nodes, the cost of naïvely calculating $A_{kl}(t_e)$ by iterating through all "at-risk" edges is nearly $n^2$. As in Section 3.1, the cost will be $O(n)$ if we instead use caching together with equation (10). In other cases, there may be restrictions on the set of edges at risk at a particular time. Here the computational burden for the naïve calculation can be substantially smaller than $O(n^2)$; yet it is generally the case that using (10) will still provide a substantial reduction in computing effort.

Our online inference algorithm during the time interval $[t_{e-1}, t_e)$ may be summarized as follows:

1. Update $\mathbf{A}(t_{e-1})$ using equation (10).
2. Compute $\hat{\mathbf{B}}(t_{e-1}) = \hat{\mathbf{B}}(t_{e-2}) + \mathbf{A}^{-1}(t_{e-1})\mathbf{W}(t_{e-1})'\Delta\mathbf{N}(t_{e-1})$.
3. Compute and cache the network statistics changed by the event $e-1$, then initialize $U_{e-1}$ with a list of those at-risk edges whose network statistics are changed by this event.
4. Compute and cache all values of network statistics changed during the time interval $[t_{e-1}, t_e)$. Define $C_{e-1}$ as the set of edges that switch to at-risk during this interval.
5. Before considering the event $e$:
   (a) Compute look-ahead summations at time $t_{e-1}$ indexed by $U_{e-1}$.
   (b) Update the covariate matrix $\mathbf{W}(t_{e-1})$ based on the cache.
   (c) Compute forward summations at time $t_e$ indexed by $U_{e-1}$ and $C_{e-1}$.

For the first event, $\mathbf{A}(t_1)$ must be initialized by naïve summation over all current at-risk edges, which requires $O(p^2 n^2)$ calculations. Assuming that the number $n$ of nodes stays roughly the same over each of the $m$ edge events, the overall computational complexity of this online inference algorithm is thus $O(p^2 n^2 + m(p^2 n + p^3))$. If a covariance matrix estimate for $\hat{\mathbf{B}}(t)$ is desired, it can also be derived online using the ideas above, since we may write it as

$$\hat{\mathbf{\Sigma}}(t) = \sum_{t_e \leq t} \mathbf{W}^-(t_e)\text{diag}\{\Delta\mathbf{N}(t_e)\}\mathbf{W}^-(t_e)^\top = \sum_{t_e \leq t} \mathbf{A}^{-1}(t_e)\big[\mathbf{W}_{ij_e}(t_e) \otimes \mathbf{W}_{ij_e}(t_e)\big]\mathbf{A}^{-1}(t_e), \quad (11)$$

where $\mathbf{W}_{ij_e}(t_e)$ denotes the vector $\mathbf{W}(t_e)^\top \Delta\mathbf{N}(t_e)$ and $\otimes$ is the outer product.

## 4 Experimental analysis

In this section, we empirically analyze the ability of our inference methods to estimate the regression coefficients as well as the predictive power of the learned models. Before discussing the experimental results, we briefly describe the synthetic and real-world data sets that we use for evaluation.

We simulate two data sets, SIM-1 and SIM-2, from ground-truth regression coefficients. In particular, we simulate a network formation process starting from time unit 0 until time 1200, where nodes arrive in the network at a constant rate $\lambda_0 = 10$ (i.e., on average, 10 nodes join the network at each time unit); the resulting simulated networks have 11,997 nodes. The edge formation process is simulated via Otaga's modified thinning algorithm [14] with an additive conditional intensity function. From time 0 to 1000, the baseline coefficient is set to $\beta_0 = 10^{-6}$; the coefficients for sender out-degree and receiver in-degree are set to $\beta_1 = \beta_4 = 10^{-7}$; the coefficients for reciprocity, transitivity, and shared contacters are set to $\beta_5 = \beta_6 = \beta_9 = 10^{-5}$; and the coefficients for sender in-degree, receiver out-degree, shared contactees, and triangle closure are set to 0. For SIM-1, these coefficients are kept constant and 118,672 edges are created. For SIM-2, between times 1000 and 1150, we increase the coefficients for transitivity and shared contacters to $\beta_6 = \beta_9 = 4 \times 10^{-5}$, and after 1150, the coefficients return to their original values; in this case, 127,590 edges are created.

We also evaluate our approach on two real-world data sets, IRVINE and METAFILTER. IRVINE is a longitudinal data set derived from an online social network of students at UC Irvine [15]. This dataset has 1,899 users and 20,296 directed contact edges between users, with timestamps for each node arrival and edge creation event. This longitudinal network spans April to October of 2004. The METAFILTER data set is from a community weblog where users can share links and discuss Web content[1]. This dataset has 51,362 users and 76,791 directed contact edges between users. The continuous-time observation spans 8/31/2007 to 2/5/2011. Note that both data sets are non-recurrent in that the creation of an edge between two nodes only occurs at most once.

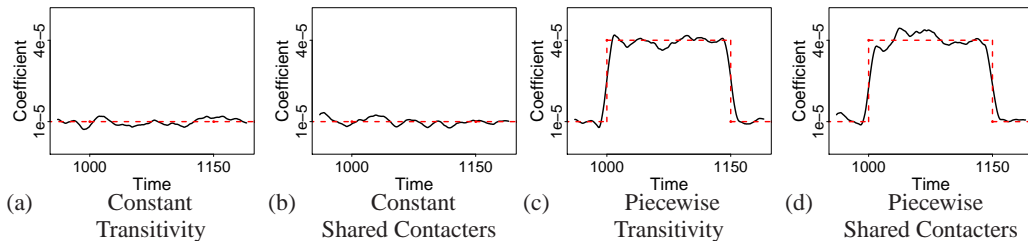

Figure 1: (a,b) Estimated time-varying coefficients on SIM-1; (c,d) Estimated time-varying coefficients on SIM-2. Ground-truth coefficients are also shown in red dashed lines.

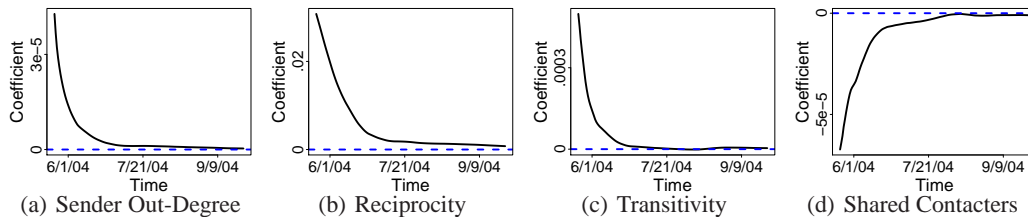

Figure 2: Estimated time-varying coefficients on IRVINE data. These plots suggest that there are two distinct phases of network evolution, consistent with an independent analysis of these data [15].

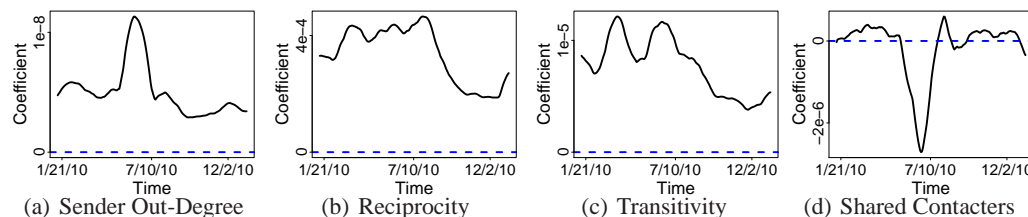

Figure 3: Estimated time-varying coefficients on METAFILTER. Here, the network effects continuously change during the observation time.

## 4.1 Recovering the time-varying regression coefficients

This section focuses on the ability of our additive Aalen modeling approach to estimate the time-varying coefficients, given an observed longitudinal network.

The first set of experiments attempts to recover the ground-truth coefficients on SIM-1 and SIM-2. We run the inference algorithm described in Section 3.3 and use an Epanechnikov smoothing kernel (with a bandwidth of 10 time units) to obtain smoothed coefficients. On SIM-1, Figures 1(a,b) show the estimated coefficients associated with the transitivity and shared contacters statistics, as well as the ground-truth coefficients. Likewise, Figures 1(c,d) show the same estimated and ground-truth coefficients for SIM-2. These results demonstrate that our inference algorithm can accurately recover the ground-truth coefficients in cases where the coefficients are fixed (SIM-1) and modulated (SIM-2). We also tried other settings for the ground-truth coefficients (e.g., multiple sinusoidal-like bumps) and found that our approach can accurately recover the coefficients in those cases as well.

On the IRVINE and METAFILTER data, we also learn time-varying coefficients which are useful for interpreting network evolution. Figure 2 shows several of the estimated coefficients for the IRVINE data, using an Epanechnikov kernel (with a bandwidth of 30 days). These coefficients suggest the existence of two distinct phases in the evolution of the network. In the first phase of network formation, the network grows at an accelerated rate. Positive coefficients for sender out-degree, reciprocity, and transitivity in these plots imply that users with a high numbers of friends tend to make more friends, tend to reciprocate their relations, and tend to make friends with their friends' friends, respectively. However, these coefficients decrease towards zero (the blue line) and enter a second phase where the network is structurally stable. Both of these phases have also been observed in an independent study of the data [15]. Figure 3 shows the estimated coefficients for METAFILTER, using an Epanechnikov kernel (with a bandwidth of 30). Interestingly, the coefficients suggest that there is a marked change in the edge formation process around 7/10/10. Unlike the IRVINE coefficients, the estimated METAFILTER coefficients continue to vary over time.

Table 1: Lengths of building, training, and test periods. The number of events are in parentheses.

| | Building | Training | Test |
|---|---|---|---|
| IRVINE | 4/15/04 – 5/11/04 (7073) | 5/12/04 – 5/31/04 (7646) | 6/1/04 – 10/19/04 (5507) |
| METAFILTER | 6/15/04 – 12/21/09 (60376) | 12/22/09 – 7/9/10 (8763) | 7/10/10 – 2/5/11 (7620) |

## 4.2 Predicting future links

We perform rolling prediction experiments over the real-world data sets to evaluate the predictive power of the learned regression models. Following the evaluation methodology of [9], we split each longitudinal data set into three periods: a statistics-building period, a training period, and a test period (Table 1). The statistics-building period is used solely to build up the network statistics, while the training period is used to learn the coefficients and the test period is used to make predictions. Throughout the training and test periods, the time-dependent statistics are continuously updated. Furthermore, for the additive Aalen model, we use the online inference technique from Section 3.3. When we predict an event in the test period, all the previous events from the test period are used as training data as well. Meanwhile, for the multiplicative Cox model, we adaptively learn the model in batch-online fashion; during the test period, for every 10 days, we retrain the model (using the Newton-Raphson technique described in Section 3.1) with additional training examples coming from the test set. Our Newton-Raphson implementation uses a step-halving procedure, halving the length of each step if necessary until $\log L(\boldsymbol{\beta})$ increases. The iterations continue until every element in $\nabla \log L(\boldsymbol{\beta})$ is smaller that $10^{-3}$ in absolute value, or until the relative increase in $\log L(\boldsymbol{\beta})$ is less than $10^{-100}$, or until 100 Newton-Raphson iterations are reached, whichever occurs first.

The baseline that we consider is logistic regression (LR) with the same time-dependent statistics used in the Aalen and Cox models. Note that logistic regression is a competitive baseline that has been used in previous link prediction studies (e.g., [11]). We learn the LR model in the same adaptive batch-online fashion as the Cox model. We also use case control sampling to address the imbalance between positive and negative cases (since at each "positive" edge event there are order of $n^2$ "negative" training cases). At each event, we sample $K$ negative training examples for that same time point. We use two settings for $K$ in the experiments: $K = 10$ and $K = 50$.

To make predictions using the additive Aalen model, one would need to extrapolate the time-varying coefficients to future time points. For simplicity, we use a uniform smoothing kernel (weighting all observations equally), with a window size of 1 or 10 days. A more advanced extrapolation technique could yield even better predictive performance for the Aalen model.

Each model can provide us with the probability of an edge formation event between two nodes at a given point in time, and so we can calculate an accumulative recall metric across all test events:

$$\text{Recall} = \frac{\sum_{(i \rightarrow j, t) \in \text{TestSet}} \mathbf{I}[j \in \text{Top}(i, t, K)]}{|\text{TestSet}|}, \tag{12}$$

where $\text{Top}(i, t, K)$ is the top-$K$ list of $i$'s potential "friends" ranked based on intensity $\lambda_{ij}(t)$.

We evaluate the predictive performance of the Aalen model (with smoothing windows of 1 and 10), the Cox model, and the LR baseline (with case control ratios 1:10 and 1:50). Figure 4(a) shows the recall results on IRVINE. In this case, both the Aalen and Cox models outperform the LR baseline; furthermore, it is interesting to note that the Aalen model with time-varying coefficients does not outperform the Cox model. One explanation for this result is that the IRVINE coefficients are pretty stable (apart from the initial phase as shown in Figure 2), and thus time-varying coefficients do not provide additional predictive power in this case. Also note that LR with ratio 1:10 outperforms 1:50. We also tried an LR ratio of 1:3 (not shown) but found that it performed nearly identically to LR 1:10; thus, both the Aalen and Cox models outperform the baseline substantially on these data.

Figure 4(b) shows the recall results on METAFILTER. As in the previous case, both the Aalen and Cox models significantly outperform the LR baseline. However, the Aalen model with time-varying coefficients also substantially outperforms the Cox model with time-fixed coefficients. In this case, estimating time-varying coefficients improves predictive performance, which makes sense because we have seen in Figure 3 that METAFILTER's coefficients tend to vary more over time. We also calculated precision results (not shown) on these data sets which confirm these conclusions.

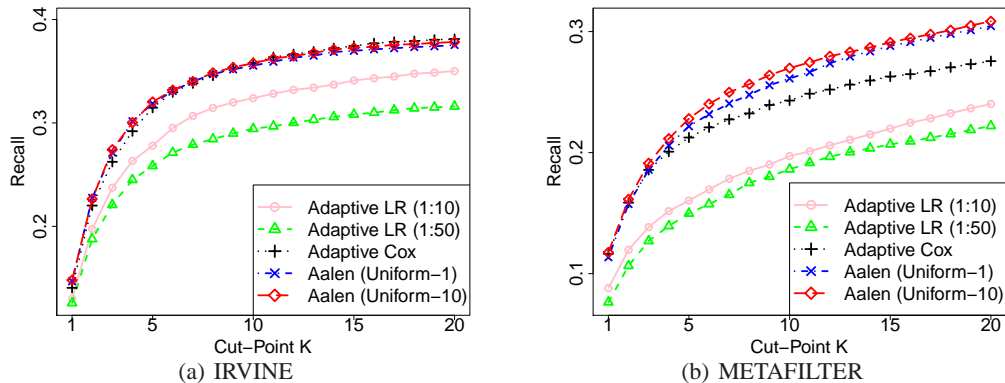

(a) IRVINE              (b) METAFILTER

Figure 4: Predictive performance of the additive Aalen model, multiplicative Cox model, and logistic regression baseline on the IRVINE and METAFILTER data sets, using recall as the metric.

## 5    Related Work and Conclusions

Evolving networks have been descriptively analyzed in exploratory fashion in a variety of domains, including email data [16], citation graphs [17], and online social networks [18]. On the modeling side, temporal versions of exponential random graph models [1, 2, 3] and latent space models [19, 4, 5, 20] have been developed. Such methods operate on cross-sectional snapshot data, while our framework models continuous-time network event data. It is worth noting that continuous-time Markov process models for longitudinal networks have been proposed previously [21]; however, these approaches have only been applied to very small networks, while our regression-based approach can scale to large networks. Recently, there has also been work on inferring unobserved time-varying networks from evolving nodal attributes which are observed [22, 23, 24]. In this paper, the main focus is the statistical modeling of observed continuous-time networks.

More recently, survival and event history models based on the Cox model have been applied to network data [8, 12, 9]. A significant difference between our previous work [9] and this paper is that scalability is achieved in our earlier work by restricting the approach to "egocentric" modeling, in which counting processes are placed only on nodes. In contrast, here we formulate scalable inference techniques for the general "relational" setting where counting processes are placed on edges. Prior work also assumed static regression coefficients, while here we develop a framework for time-varying coefficients for the additive Aalen model. Regression models with varying coefficients have been previously proposed in other contexts [25], including a time-varying version of the Cox model [26], although to the best of our knowledge such models have not been developed or fitted on longitudinal networks.

A variety of link prediction techniques have also been investigated by the machine learning community over the past decade (e.g., [27, 28, 29]). Many of these methods use standard classifiers (such as logistic regression) and take advantage of key features (such as similarity measures among nodes) to make accurate predictions. While our focus is not on feature engineering, we note that arbitrary network and nodal features such as those developed for link prediction can be incorporated into our continuous-time regression framework. Other link prediction techniques based on matrix factorization [30] and random walks [11] have also been studied. While these link prediction techniques mainly focus on making accurate predictions, our proposed approach here not only gives accurate predictions but also provides a statistical model (with time-varying coefficient estimates) which can be useful in evaluating scientific hypotheses.

In summary, we have developed multiplicative and additive regression models for large-scale continuous-time longitudinal networks. On simulated and real-world data, we have shown that the proposed inference approach can accurately estimate regression coefficients and that the learned model can be used for interpreting network evolution and predicting future network events. An interesting direction for future work would be to incorporate time-dependent nodal attributes (such as textual content) into this framework and to investigate regularization methods for these models.

### Acknowledgments
This work is supported by ONR under the MURI program, Award Number N00014-08-1-1015.

## Footnotes

[1]The METAFILTER data are available at `http://mssv.net/wiki/index.php/Infodump`

# References

[1] S. Hanneke and E. P. Xing. Discrete temporal models of social networks. In *Proc. 2006 Conf. on Statistical Network Analysis*, pages 115–125. Springer-Verlag, 2006.

[2] D. Wyatt, T. Choudhury, and J. Bilmes. Discovering long range properties of social networks with multi-valued time-inhomogeneous models. In *Proc. 24th AAAI Conf. on AI*, 2010.

[3] P. N. Krivitsky and M. S. Handcock. A separable model for dynamic networks. *Under review*, November 2010. http://arxiv.org/abs/1011.1937.

[4] W. Fu, L. Song, and E. P. Xing. Dynamic mixed membership blockmodel for evolving networks. In *Proc. 26th Intl. Conf. on Machine Learning*, pages 329–336. ACM, 2009.

[5] J. Foulds, C. DuBois, A. Asuncion, C. Butts, and P. Smyth. A dynamic relational infinite feature model for longitudinal social networks. In *AI and Statistics*, volume 15 of *JMLR W&C Proceedings*, pages 287–295, 2011.

[6] P. K. Andersen, O. Borgan, R. D. Gill, and N. Keiding. *Statistical Models Based on Counting Processes*. Springer, 1993.

[7] O. O. Aalen, O. Borgan, and H. K. Gjessing. *Survival and Event History Analysis: A Process Point of View*. Springer, 2008.

[8] C. T. Butts. A relational event framework for social action. *Soc. Meth.*, 38(1):155–200, 2008.

[9] D. Q. Vu, A. U. Asuncion, D. R. Hunter, and P. Smyth. Dynamic egocentric models for citation networks. In *Proc. 28th Intl. Conf. on Machine Learning*, pages 857–864, 2011.

[10] P. Holland and S. Leinhardt. A dynamic model for social networks. *J. Math. Soc.*, 5:5–20, 1977.

[11] L. Backstrom and J. Leskovec. Supervised random walks: Predicting and recommending links in social networks. In *Proceedings of the 4th ACM International Conference on Web Search and Data Mining*, pages 635–644. ACM, 2011.

[12] P. O. Perry and P. J. Wolfe. Point process modeling for directed interaction networks. *Under review*, October 2011. http://arxiv.org/abs/1011.1703.

[13] D. R. Cox. Regression models and life-tables. *J. Roy. Stat. Soc., Series B*, 34:187–220, 1972.

[14] D. J. Daley and D. Vere-Jones. *An Introduction to the Theory of Point Processes, Volume 1*. Probability and its Applications (New York). Springer, New York, 2nd edition, 2008.

[15] P. Panzarasa, T. Opsahl, and K. M. Carley. Patterns and dynamics of users' behavior and interaction: Network analysis of an online community. *J. Amer. Soc. for Inf. Sci. and Tech.*, 60(5):911–932, 2009.

[16] G. Kossinets and D. J. Watts. Empirical analysis of an evolving social network. *Science*, 311(5757):88–90, 2006.

[17] J. Leskovec, J. Kleinberg, and C. Faloutsos. Graphs over time: densification laws, shrinking diameters and possible explanations. In *Proc. 11th ACM SIGKDD Intl. Conf. on Knowledge Discovery in Data Mining*, pages 177–187. ACM, 2005.

[18] B. Viswanath, A. Mislove, M. Cha, and K. P. Gummadi. On the evolution of user interaction in Facebook. In *Proc. 2nd ACM SIGCOMM Wkshp. on Social Networks*, pages 37–42. ACM, 2009.

[19] P. Sarkar and A. Moore. Dynamic social network analysis using latent space models. *SIGKDD Explorations*, 7(2):31–40, 2005.

[20] Q. Ho, L. Song, and E. Xing. Evolving cluster mixed-membership blockmodel for time-varying networks. In *AI and Statistics*, volume 15 of *JMLR W&C Proceedings*, pages 342–350, 2011.

[21] T. A. B. Snijders. Models for longitudinal network data. *Mod. Meth. in Soc. Ntwk. Anal.*, pages 215–247, 2005.

[22] S. Zhou, J. Lafferty, and L. Wasserman. Time varying undirected graphs. *Machine Learning*, 80:295–319, 2010.

[23] A. Ahmed and E. P. Xing. Recovering time-varying networks of dependencies in social and biological studies. *Proc. Natl. Acad. Scien.*, 106(29):11878–11883, 2009.

[24] M. Kolar, L. Song, A. Ahmed, and E. P. Xing. Estimating time-varying networks. *Ann. Appl. Stat.*, 4(1):94–123, 2010.

[25] Z. Cai, J. Fan, and R. Li. Efficient estimation and inferences for varying-coefficient models. *J. Amer. Stat. Assn.*, 95(451):888–902, 2000.

[26] T. Martinussen and T.H. Scheike. *Dynamic Regression Models for Survival Data*. Springer, 2006.

[27] D. Liben-Nowell and J. Kleinberg. The link-prediction problem for social networks. *J. Amer. Soc. for Inf. Sci. and Tech.*, 58(7):1019–1031, 2007.

[28] M. Al Hasan, V. Chaoji, S. Salem, and M. Zaki. Link prediction using supervised learning. In *SDM '06: Workshop on Link Analysis, Counter-terrorism and Security*, 2006.

[29] J. Leskovec, D. Huttenlocher, and J. Kleinberg. Predicting positive and negative links in online social networks. In *Proc. 19th Intl. World Wide Web Conference*, pages 641–650. ACM, 2010.

[30] D. M. Dunlavy, T. G. Kolda, and E. Acar. Temporal link prediction using matrix and tensor factorizations. *ACM Transactions on Knowledge Discovery from Data*, 5(2):10, February 2011.

